# Modeling Complex Cells in an Awake Macaque During Natural Image Viewing

**William E. Vinje**
vinje@socrates.berkeley.edu

Department of Molecular and
Cellular Biology, Neurobiology Division
University of California, Berkeley
Berkeley, CA, 94720

**Jack L. Gallant**
gallant@socrates.berkeley.edu

Department of Psychology
University of California, Berkeley
Berkeley, CA, 94720

## Abstract

We model the responses of cells in visual area V1 during natural vision. Our model consists of a classical energy mechanism whose output is divided by nonclassical gain control and texture contrast mechanisms. We apply this model to *review movies*, a stimulus sequence that replicates the stimulation a cell receives during free viewing of natural images. Data were collected from three cells using five different review movies, and the model was fit separately to the data from each movie. For the energy mechanism alone we find modest but significant correlations ($r_E = 0.41, 0.43, 0.59, 0.35$) between model and data. These correlations are improved somewhat when we allow for suppressive surround effects ($r_{E+G} = 0.42, 0.56, 0.60, 0.37$). In one case the inclusion of a delayed suppressive surround dramatically improves the fit to the data by modifying the time course of the model's response.

## 1 INTRODUCTION

Complex cells in the primary visual cortex (area V1 in primates) are tuned to localized visual patterns of a given spatial frequency, orientation, color, and drift direction (De Valois & De Valois, 1990). These cells have been modeled as linear spatio-temporal filters whose output is rectified by a static nonlinearity (Adelson & Bergen, 1985); more recent models have also included a divisive contrast gain control mechanism (Heeger, 1992; Wilson & Humanski, 1993; Geisler & Albrecht, 1997). We apply a modified form of these models to a stimulus that simulates natural vision. Our model uses relatively few parameters yet incorporates the cells' temporal response properties and suppressive influences from beyond the classical receptive field ($CRF$).

## 2 METHODS

**Data Collection:** Data were collected from one awake behaving Macaque monkey, using single unit recording techniques described elsewhere (Connor *et al.*, 1997).[1] First, the cell's receptive field size and location were estimated manually, and tuning curves were objectively characterized using two-dimensional sinusoidal gratings. Next a static color image of a natural scene was presented to the animal and his eye position was recorded continuously as he freely scanned the image for 9 seconds (Gallant *et al.*, 1998).[2] Image patches centered on the position of the cell's $CRF$ (and 2-4 times the $CRF$ diameter) were then extracted using an automated procedure. The sequence of image patches formed a continuous 9 second *review movie* that simulated all of the stimulation that had occurred in and around the $CRF$ during free viewing.[3] Although the original image was static, the review movies contain the temporal dynamics of the saccadic eye movements made by the animal during free viewing. Finally, the review movies were played in and around the $CRF$ while the animal performed a fixation task.

During free viewing each eye position is unique, so each image patch is likely to enter the $CRF$ only once. The review movies were therefore replayed several times and the cell's average response with respect to the movie timestream was computed from the peri-stimulus time histogram (PSTH). These review movies also form the model's stimulus input, while its output is relative spike probability versus time (the model cell's PSTH).

Before applying the model each review movie was preprocessed by converting to gray scale (since the model does not consider color tuning), setting the average luminance level to zero (on a frame by frame basis) and prefiltering with the human contrast sensitivity function to more accurately reflect the information reaching cells in V1.

**Divisive Normalization Model:** The model consists of a classical receptive field energy mechanism, $E_{CRF}$, whose output is divided by two nonclassical suppressive mechanisms, a gain control field, $G$, and a texture contrast field, $T$.

$$PSTH_{model}(t) \propto \frac{E_{CRF}(t)}{1 + \alpha\, G(t - \delta) + \beta\, T(t - \delta)} \tag{1}$$

We include a delay parameter for suppressive effects, consistent with the hypothesis that these effects may be mediated by local cortical interactions (Heeger, 1992; Wilson & Humanski, 1993). Any latency difference between the central energy mechanism and the suppressive surround will be reflected as a positive delay offset ($\delta > 0$ in Equation 1).

**Classical Receptive Field Energy Mechanism:** The energy mechanism, $E_{CRF}$, is composed of four phase-dependent subunits, $U^\phi$. Each subunit computes an inner product in space and a convolution in time between the model cell's space-time classical receptive field, $CRF^\phi(x, y, \tau)$, and the image, $I(x, y, t)$.

$$U^\phi(t) = \iiint CRF^\phi(x, y, \tau) \cdot I(x, y, t - \tau)\, dx\, dy\, d\tau \tag{2}$$

The model presented here incorporates the simplifying assumption of a space-time separable receptive field structure, $CRF^\phi(x, y, \tau) = CRF^\phi(x, y) \, CRF(\tau)$.

$$U^\phi(t) = \sum_\tau CRF(\tau) \left( \sum_x \sum_y CRF^\phi(x, y) \cdot I(x, y, t - \tau) \right) \qquad (3)$$

Time is discretized into frames and space is discretized into pixels that match the review movie input. $CRF^\phi(x, y)$ is modeled as a sinusoidal grating that is spatially weighted by a Gaussian envelope (i.e. a Gabor function). In this paper $CRF(\tau)$ is approximated as a delta function following a constant latency. This minimizes model parameters and highlights the model's responses to the stimulus present at each fixation. The latency, orientation and spatial frequency of the grating, and the size of the $CRF$ envelope, are all determined empirically by maximizing the fit between model and data.[4]

A static non-linearity ensures that the model PSTH does not become negative. We have examined both half-wave rectification, $\tilde{U}^\phi(t) = \max[U^\phi(t), 0]$, and half-squaring, $\tilde{U}^\phi(t) = (\max[U^\phi(t), 0])^2$; here we present the results from half-wave rectification. Half-squaring produces small changes in the model PSTH but does not improve the fit to the data.

The energy mechanism is made phase invariant by averaging over the rectified phase-dependent subunits:

$$E_{CRF}(t) = \frac{1}{4} \left( \tilde{U}^0(t) + \tilde{U}^{90}(t) + \tilde{U}^{180}(t) + \tilde{U}^{270}(t) \right) \qquad (4)$$

**Gain Control Field:** Cells in V1 incorporate a contrast gain control mechanism that compensates for changes in local luminance. The gain control field, $G$, models this effect as the total image power in a region encompassing the $CRF$ and surround.

$$G(t - \delta) = \sum_\tau CRF(\tau) \left( \sum_{k_x} \sum_{k_y} \sqrt{P(k_x, k_y, \tau)} \right) \qquad (5)$$

$$P(k_x, k_y, \tau) = FFT[\mu_G(x, y, \tau)] \, FFT^*[\mu_G(x, y, \tau)] \qquad (6)$$

$$\mu_G(x, y, \tau) = \nu_G(x, y) \, I(x, y, (t - \delta) - \tau) \qquad (7)$$

$P(k_x, k_y, \tau)$ is the spatial Fourier power of $\mu_G(x, y, \tau)$ and $\nu_G$ is a two dimensional Gaussian weighting function whose width sets the size of the gain control field.

Heeger's (1992) divisive gain control term sums over many discrete energy mechanisms that tile space in and around the area of the $CRF$. Equation 5 approximates Heeger's approach in the limiting case of dense tiling.

**Texture Contrast Field:** Cells in area V1 can be affected by the image surrounding the region of the $CRF$ (Knierim & Van Essen, 1992). The responses of many V1 cells are highest when the optimal stimulus is presented alone within the $CRF$, and lowest when that stimulus is surrounded with a texture of similar orientation and frequency. The texture contrast field, $T$, models this effect as the image power

in the spatial region surrounding the $CRF$ that matches the $CRF$'s orientation and spatial frequency.

$$T(t - \delta) = \frac{1}{4} \sum_{\phi=0}^{90,180,270} \left[ \sum_\tau CRF(\tau) \left( \sum_{k_x} \sum_{k_y} \sqrt{P^\phi(k_x, k_y, \tau)} \right) \right] \qquad (8)$$

$$P^\phi(k_x, k_y, \tau) = FFT[\mu_T^\phi(x, y, \tau)] \, FFT^*[\mu_T^\phi(x, y, \tau)] \qquad (9)$$

$$\mu_T^\phi(x, y, \tau) = \xi_T^\phi(x, y) \, (1 - \nu_{CRF}(x, y)) \, I(x, y, (t - \delta) - \tau) \qquad (10)$$

$\xi_T^\phi$ is a Gabor function whose orientation and spatial frequency match those of the best fit $CRF^\phi(x, y)$. The envelope of $\xi_T^\phi$ defines the size of the texture contrast field. $\nu_{CRF}$ is a two dimensional Gaussian weighting function whose width matches the $CRF$ envelope, and which suppresses the image center. Thus the texture contrast term picks up oriented power from an annular region of the image surrounding the $CRF$ envelope. $T$ is made phase invariant by averaging over phase.

## 3   RESULTS

Thus far our model has been evaluated on a small data set collected as part of a different study (Gallant *et al.*, 1998). Two cells, 87A and 98C, were examined with one review movie each, while cell 97A was examined with three review movies. Using this data set we compare the model's response in two interesting situations: cell 97A, which had high orientation-selectivity, versus cell 87A, which had poor orientation-selectivity; and cell 98C, which was directionally-selective, versus cell 97A, which was not directionally-selective.

**CRF Energy Mechanism:** We separately fit the energy mechanism parameters to each of the three different cells. For cell 97A the three review movies were fit independently to test for consistency of the best fit parameters.

Table 1 shows the correlation between model and data using only the $CRF$ energy mechanism ($\alpha = \beta = 0$ in Equation 1). The significance of the correlations was assessed via a permutation test. The correlation values for cells 97A and 98C, though modest, are significant ($p < 0.01$). For these cells the 95% confidence intervals on the best fit parameter values are consistent with estimates from the flashed grating tests. The best fit parameter values for cell 97A are also consistent across the three independently fit review movies.

The model best accounts for the data from cell 97A. This cell was highly selective for vertical gratings and was not directionally-selective. Figure 1 compares the PSTH obtained from cell 97A with movie B to the model PSTH. The model generally responds to the same features that drive the real cell, though the match is imperfect. Much of the discrepancy between the model and data arises from our approximation of $CRF(t)$ as a delta function. The model's response is roughly constant during

| Cell | 87A | 97A | 97A | 97A | 98C |
|---|---|---|---|---|---|
| Movie | A | A | B | C | A |
| Oriented | No | Yes | Yes | Yes | Yes |
| Directional | No | No | No | No | Yes |
| $r_E$ | **NA** | **0.41** | **0.43** | **0.59** | **0.35** |

Table 1: Correlations between model and data PSTHs. *Oriented* cells showed orientation-selectivity in the flashed grating test while *Directional* cells showed directional-selectivity during manual characterization. $r_E$ is the correlation between $E_{CRF}$ and the data. No fit was obtained for cell 87A.

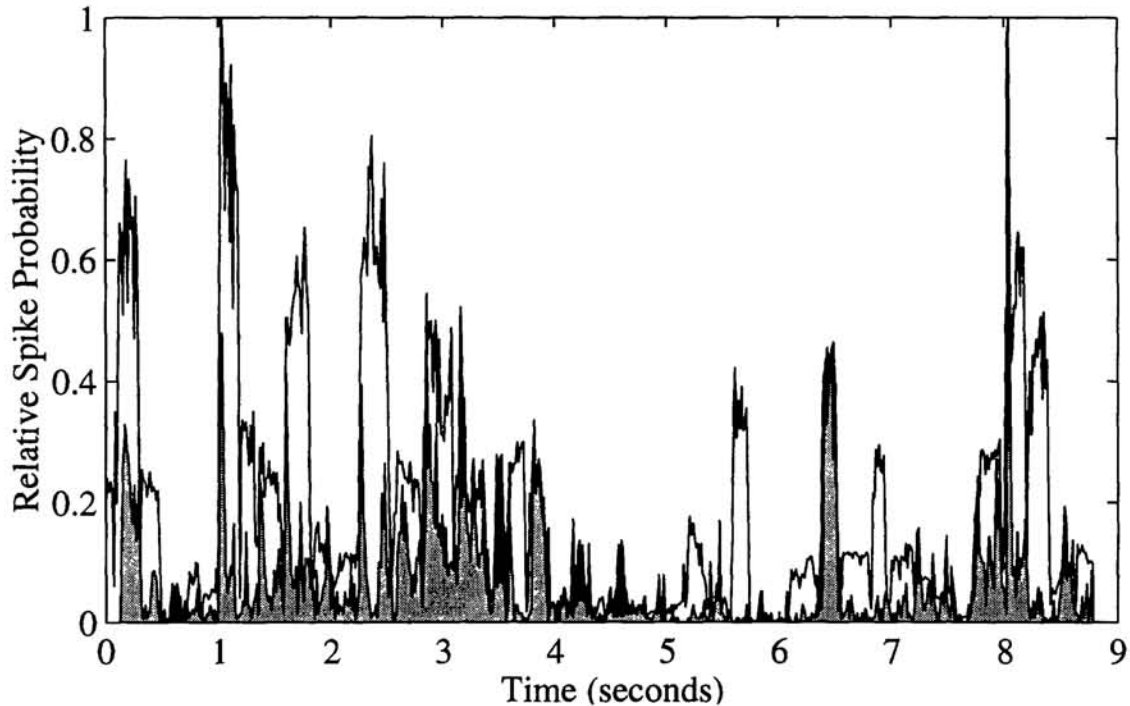

Figure 1: *CRF* energy mechanism versus data (Cell 97A, Movie B). White indicates that the model response is greater than the data, while black indicates the data is greater than the model and gray indicates regions of overlap. A perfect match between model and data would result in the entire area under the curve being gray. Our approximation of $CRF(t)$ leads to a relatively constant model PSTH during each fixation. In contrast the real cell generally gives a phasic response as each saccade brings a new stimulus into the CRF. In general the same movie features drive both model and cell.

each fixation, which causes the model PSTH to appear stepped. In contrast the data PSTH shows a strong phasic response at the beginning of each fixation when a new stimulus patch enters the cell's $CRF$.

The model is less successful at accounting for the responses of the directionally-selective cell, 98C. This is probably because the model's space-time separable receptive field misses motion energy cues that drive the cell. The model completely failed to fit the data from cell 87A. This cell was not orientation-selective, so the fitting procedure was unable to find an appropriate orientation for the $CRF^\phi(x, y)$ Gabor function.[5]

**CRF Energy Mechanism with Suppressive Surround:** Table 2 lists the improvements in correlation obtained by adding the gain control term ($\alpha > 0, \beta = 0$ in Equation 1). For cell 97A (all three movies) the best correlations are obtained when the surround effects are delayed by 56 ms relative to the center. The best correlation for cell 98C is obtained when the surround is not delayed.

In three out of four cases the correlation values are barely improved when the surround effects are included, suggesting that the cells were not strongly surround-inhibited by these review movies. However, the improvement is quite striking in the

| Cell | 97A | 97A | 97A | 98C |
|---|---|---|---|---|
| Movie | A | B | C | A |
| $r_{E+G}$ | 0.42 | 0.56 | 0.60 | 0.37 |
| $\Delta r$ | +0.01 | +0.13 | +0.01 | +0.02 |

Table 2: Correlation improvements due to surround gain control mechanism. $r_{E+G}$ gives the correlation value between the best fit model and the data. $\Delta r$ gives the improvement over $r_E$. Including $G$ in Equation 1 leads to a dramatic correlation increase for cell 97A, movie B, but not for the other review movies.

case of cell 97A, movie B. Figure 2 compares the data with a model using both $E_{crf}$ and $G$ in Equation 1. Here the delayed surround suppresses the sustained responses seen in Figure 1 and results in a more phasic model PSTH that closely matches the data.

We consider $G$ and $T$ fields both independently and in combination. For each we independently fit for $\alpha$, $\beta$, $\delta$, and the size of the suppressive fields. However, the oriented Fourier power correlates with the total Fourier power for our sample of natural images, so that $G$ and $T$ are highly correlated. Combined fitting of $G$ and $T$ terms leads to competition and dominance by $G$ (i.e. $\beta \to 0$). In this paper we only report the effects of the gain control mechanism; the texture contrast mechanism results in similar (though slightly degraded) results.

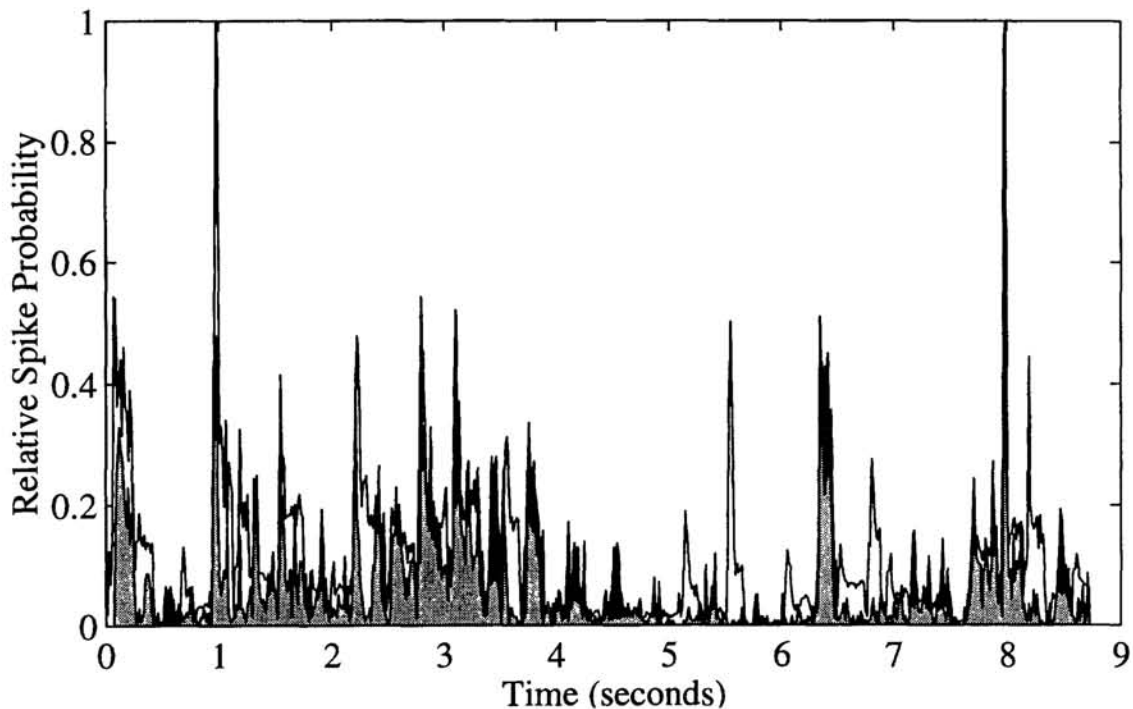

Figure 2: $CRF$ energy mechanism with delayed surround gain control versus data (Cell 97A, Movie B). Color scheme as in Figure 1. The inclusion of the delayed $G$ term results in a more phasic model response which greatly improves the match between model and data.

## 4  DISCUSSION

This preliminary study suggests that models of the form outlined here show great promise for describing the responses of area V1 cells during natural vision. For comparison consider the correlation values obtained from an earlier neural network model that attempted to reproduce V1 cells' responses to a variety of spatial patterns (Lehky *et al.* 1992). They report a median correlation value of 0.65 for complex stimuli, whereas the average correlation score from Table 2 is 0.49. This is remarkable considering that our model has only 7 free parameters, a very limited data set for fitting, doesn't yet consider color tuning or directional-selectivity *and* considers response across time.

Future implementations of the model will use a more sophisticated energy mechanism that allows for nonseparable space time receptive field structure and more realistic temporal response dynamics. We will also incorporate more detail into the surround mechanisms, such as asymmetric surround structure and a broadband texture contrast term.

By abstracting physiological observation into approximate functional forms our model balances explanatory power against parametric complexity. A cascaded series of these models may form the foundation for future modeling of cells in extra-striate areas V2 and V4. Natural image stimuli may provide an appropriate stimulus set for development and validation of these extrastriate models.

### Acknowledgements

We thank Joseph Rogers for assistance in this study, Maneesh Sahani for the extremely useful suggestion of fitting the CRF parameters, Charles Connor for help with data collection and David Van Essen for support of data collection.

## Footnotes

[1]Recording was performed under a university-approved protocol and conformed to all relevant NIH and USDA guidelines.

[2]Images were taken from a Corel Corporation photo-CD library at 1280x1024 resolution.

[3]Eye position data were collected at 1 KHz, whereas the monitor display rate was 72.5 Hz (14 ms per frame). Therefore each review movie frame was composed of the average stimulation occurring during the corresponding 13.8 ms of free viewing.

[4] As a fit statistic we use the linear correlation coefficient (Pearson's $r$) between model and data. Fitting is done with a gradient ascent algorithm. Our choice of correlation as a statistic eliminates the need to explicitly consider model normalization as a variable, and is very sensitive to latency mismatches between model and data. However, linear correlation is more prone to noise contamination than is $\chi^2$.

[5]For cell 87A the correlation values in the orientation and spatial frequency parameter subspace contained three roughly equivalent maxima. Contamination by multiple cells was unlikely due to this cell's excellent isolation.

### References

Adelson, E. H. & Bergen, J. R. (1985) Spatiotemporal energy models for the perception of motion. *Journal of the Optical Society of America, A*, **2**, 284-299.

Connor, C. C., Preddie, D. C., Gallant, J. L. & Van Essen, D. C. (1997) Spatial attention effects in macaque area V4. *Journal of Neuroscience*, **77**, 3201-3214.

De Valois, R. L. & De Valois, K. K. (1990) *Spatial Vision*. New York: Oxford University Press.

Gallant, J. L., Connor, C. E., & Van Essen, D. C. (1998) Neural Activity in Areas V1, V2 and V4 During Free Viewing of Natural Scenes Compared to Controlled Viewing. *NeuroReport*, **9**.

Geisler, W. S., Albrecht, D. G. (1997) Visual cortex neurons in monkeys and cats: Detection, discrimination, and identification. *Visual Neuroscience*, **14**, 897-919.

Heeger, D. J. (1992) Normalization of cell responses in cat striate cortex. *Visual Neuroscience*, **9**, 181-198.

Knierim, J. J. & Van Essen, D. C. (1992) Neuronal responses to static texture patterns in area V1 of the alert macaque monkey. *Journal of Neurophysiology*, **67**, 961-980.

Lehky, S. R., Sejnowski, T. J. & Desimone, R. (1992) Predicting Responses of Nonlinear Neurons in Monkey Striate Cortex to Complex Patterns. *Journal of Neuroscience*, **12**, 3568-3581.

Wilson, H. R. & Humanski, R. (1993) Spatial frequency adaptation and contrast gain control. *Vision Research*, **33**, 1133-1149.